# Selective Labeling via Error Bound Minimization

**Quanquan Gu**[†]**, Tong Zhang**[‡]**, Chris Ding**[§]**, Jiawei Han**[†]
[†]Department of Computer Science, University of Illinois at Urbana-Champaign
[‡]Department. of Statistics, Rutgers University
[§]Department. of Computer Science & Engineering, University of Texas at Arlington
qgu3@illinois.edu, tzhang@stat.rutgers.edu, chqding@uta.edu, hanj@cs.uiuc.edu

## Abstract

In many practical machine learning problems, the acquisition of labeled data is often expensive and/or time consuming. This motivates us to study a problem as follows: given a label budget, how to select data points to label such that the learning performance is optimized. We propose a selective labeling method by analyzing the out-of-sample error of Laplacian regularized Least Squares (LapRLS). In particular, we derive a deterministic out-of-sample error bound for LapRLS trained on subsampled data, and propose to select a subset of data points to label by minimizing this upper bound. Since the minimization is a combinational problem, we relax it into continuous domain and solve it by projected gradient descent. Experiments on benchmark datasets show that the proposed method outperforms the state-of-the-art methods.

## 1   Introduction

The performance of (semi-)supervised learning methods typically depends on the amount of labeled data. Roughly speaking, the more the labeled data, the better the learning performance will be. However, in many practical machine learning problems, the acquisition of labeled data is often expensive and/or time consuming. To overcome this problem, *active learning* [9, 10] was proposed, which iteratively queries the oracle (labeler) to obtain the labels at new data points. Representative methods include support vector machine (SVM) active learning [19, 18], *agnostic* active learning [2, 5, 14], etc. Due to the close interaction between the learner and the oracle, active learning can be advantageous to achieve better learning performance. Nevertheless, in many real-world applications, such an interaction may not be feasible. For example, when one turns to Amazon Mechanical Turk[1] to label data, the interaction between the learner and the labeling workers is very limited. Therefore, standard active learning is not very practical in this case.

Another potential solution to the label deficiency problem is *semi-supervised learning* [7, 22, 21, 4], which aims at combining a small number of labeled data and a large amount of unlabeled data to improve the learning performance. In a typical setting of semi-supervised learning, a small set of labeled data is assumed to be given at hand or randomly generated in practice. However, randomly selecting (uniformly sampling) data points to label is unwise because not all the data points are equally informative. It is desirable to obtain a labeled subset which is most beneficial for semi-supervised learning.

In this paper, based on the above motivation, we investigate a problem as follows: given a fixed label budget, how to select a subset of data points to label such that the learning performance is optimized. We refer to this problem as *selective labeling*, in contrast to conventional random labeling. To achieve the goal of selective labeling, it is crucial to consider the out-of-sample error of a specific learner. We choose Laplacian Regularized Least Squares (LapRLS) as the learner [4] because it is a

state-the-art semi-supervised learning method, and takes many linear regression methods as special cases (e.g., ridge regression [15]). We derive a deterministic out-of-sample error bound for LapRLS trained on subsampled data, which suggests to select the data points to label by minimizing this upper bound. The resulting selective labeling method is a combinatorial optimization problem. In order to optimize it effectively and efficiently, we relax it into a continuous optimization problem, and solve it by projected gradient descent algorithm followed by discretization. Experiments on benchmark datasets show that the proposed method outperforms the state-of-the-art methods.

The remainder of this paper is organized as follows. In Section 2, we briefly review manifold regularization and LapRLS. In Section 3, we derive an out-of-sample error bound for LapRLS on subsampled data, and present a selective labeling criterion by minimizing the this bound, followed by its optimization algorithm. We discuss the connections between the proposed method and several existing experimental design approaches in Section 4. The experiments are demonstrated in Section 5. We conclude this paper in Section 6.

## 2 Review of Laplacian Regularized Least Squares

Given a data set $\{(\mathbf{x}_1, y_1), \ldots, (\mathbf{x}_n, y_n)\}$ where $\mathbf{x}_i \in \mathbf{R}^d$ and $y_i \in \{\pm 1\}$, Laplacian Regularized Least Squares (LapRLS) [4] aims to learn a linear function $f(\mathbf{x}) = \mathbf{w}^T \mathbf{x}$. In order to estimate and preserve the geometrical and topological properties of the data, LapRLS [4] assumes that if two data points $\mathbf{x}_i$ and $\mathbf{x}_j$ are close in the intrinsic geometry of the data distribution, the labels of this two points are also close to each other. Let $f(\mathbf{x})$ be a function that maps the original data point $\mathbf{x}$ in a compact submanifold $\mathcal{M}$ to $\mathbb{R}$, we use $||f||^2_{\mathcal{M}} = \int_{\mathbf{x} \in \mathcal{M}} || \bigtriangledown_{\mathcal{M}} f ||^2 d\mathbf{x}$ to measure the smoothness of $f$ along the geodesics in the intrinsic manifold of the data, where $\bigtriangledown_{\mathcal{M}} f$ is the gradient of $f$ along the manifold $\mathcal{M}$. Recent study on spectral graph theory [8] has demonstrated that $||f||^2_{\mathcal{M}}$ can be discretely approximated through a nearest neighbor graph on a set of data points. Given an affinity matrix $\mathbf{W} \in \mathbb{R}^{n \times n}$ of the graph, $||f||^2_{\mathcal{M}}$ is approximated as:

$$||f||^2_{\mathcal{M}} \approx \frac{1}{2} \sum_{ij} ||f_i - f_j||^2_2 W_{ij} = \mathbf{f}^T \mathbf{L} \mathbf{f}, \tag{1}$$

where $f_i$ is a shorthand for $f(\mathbf{x}_i)$, $\mathbf{f} = [f_1, \ldots, f_n]^T$, $\mathbf{D}$ is a diagonal matrix, called degree matrix, with $D_{ii} = \sum_{j=1}^n W_{ij}$, and $\mathbf{L} = \mathbf{D} - \mathbf{W}$ is the combinatorial graph Laplacian [8]. Eq. (1) is called *Manifold Regularization*. Intuitively, the regularization incurs a heavy penalty if neighboring points $\mathbf{x}_i$ and $\mathbf{x}_j$ are mapped far apart.

Based on manifold regularization, LapRLS solves the following optimization problem,

$$\arg \min_{\mathbf{w}} ||\mathbf{X}^T \mathbf{w} - \mathbf{y}||^2_2 + \frac{\lambda_A}{2} ||\mathbf{w}||^2_2 + \frac{\lambda_I}{2} \mathbf{w}^T \mathbf{X} \mathbf{L} \mathbf{X}^T \mathbf{w}, \tag{2}$$

where $\lambda_A, \lambda_I > 0$ are positive regularization parameters, $\mathbf{X} = [\mathbf{x}_1, \ldots, \mathbf{x}_n]$ is the design matrix, $\mathbf{y} = [y_1, \ldots, y_n]^T$ is the response vector, $||\mathbf{w}||_2$ is $\ell_2$ regularization of linear function, and $\mathbf{w}^T \mathbf{X} \mathbf{L} \mathbf{X}^T \mathbf{w}$ is manifold regularization of $f(\mathbf{x}) = \mathbf{w}^T \mathbf{x}$. When $\lambda_I = 0$, LapRLS reduces to ridge regression [15]. A bias term $b$ can be incorporated into the form by expanding the weight vector and input feature vector as $\mathbf{w} \leftarrow [\mathbf{w}; b]$ and $\mathbf{x} \leftarrow [\mathbf{x}; 1]$. Note that Eq. (2) is a supervised version of LapRLS, because only labeled data are used in manifold regularization. Although our derivations are based on this version in the rest of the paper, the results can be extended to semi-supervised version of LapRLS straightforwardly.

## 3 The Proposed Method

### 3.1 Problem Formulation

The generic problem of selective labeling is as follows. Given a set of data points $\mathcal{X} = \{\mathbf{x}_1, \ldots, \mathbf{x}_n\}$, namely the pool of candidate data points, our goal is to find a subsample $\mathcal{L} \subset \{1, \ldots, n\}$, which contains the most informative $|\mathcal{L}| = l$ points.

To derive a selective labeling approach for LapRLS, we first derive an out-of-sample error bound of LapRLS.

## 3.2 Out-of-Sample Error Bound of LapRLS

We define the function class of LapRLS as follows.

**Definition 1.** *The function class of LapRLS is $\mathcal{F}_B = \{\mathbf{x} \to \mathbf{w}^T\mathbf{x} \mid \lambda_A ||\mathbf{w}||_2^2 + \lambda_I \mathbf{w}^T \mathbf{X}\mathbf{L}\mathbf{X}^T\mathbf{w} \leq B\}$, where $\mathbf{X} = [\mathbf{x}_1, \ldots, \mathbf{x}_n]$, and $B > 0$ is a constant.*

Consider the following linear regression model,

$$\mathbf{y} = \mathbf{X}^T\mathbf{w}^* + \boldsymbol{\epsilon}, \tag{3}$$

where $\mathbf{X} = [\mathbf{x}_1, \ldots, \mathbf{x}_n]$ is the design matrix, $\mathbf{y} = [y_1, \ldots, y_n]^T$ is the response vector, $\mathbf{w}^*$ is the true weight vector which is unknown, and $\boldsymbol{\epsilon} = [\epsilon_1, \ldots, \epsilon_n]^T$ is the noise vector with $\epsilon_i$ an unknown noise with zero mean. We assume that different observations have noises that are independent, but with equal variance $\sigma^2$.

Moreover, we assume that the true weight vector $\mathbf{w}^*$ satisfies

$$\lambda_A ||\mathbf{w}^*||_2^2 + \lambda_I (\mathbf{w}^*)^T \mathbf{X}\mathbf{L}\mathbf{X}^T\mathbf{w}^* \leq B, \tag{4}$$

which implies that the true hypothesis belongs to the function class of LapRLS in Definition 1. In this case, the approximation error vanishes and the excess error equals to the estimation error. Note that this assumption can be relaxed with more effort, under which we can derive a similar error bound as below. For simplicity, the following derivations are built upon the assumption in Eq. (4).

In selective labeling, we are interested in estimating $\mathbf{w}^*$ using LapRLS in Eq. (2) from a subsample $\mathcal{L} \in \{1, \ldots, n\}$. Denote the subsample of $\mathbf{X}$ by $\mathbf{X}_\mathcal{L}$, the subsample of $\mathbf{y}$ by $\mathbf{y}_\mathcal{L}$, and the subsample of $\boldsymbol{\epsilon}$ by $\boldsymbol{\epsilon}_\mathcal{L}$. The solution of LapRLS is given by

$$\hat{\mathbf{w}}_\mathcal{L} = (\mathbf{X}_\mathcal{L}\mathbf{X}_\mathcal{L}^T + \lambda_A \mathbf{I} + \lambda_I \mathbf{X}_\mathcal{L}\mathbf{L}_\mathcal{L}\mathbf{X}_\mathcal{L}^T)^{-1}\mathbf{X}_\mathcal{L}\mathbf{y}_\mathcal{L}, \tag{5}$$

where $\mathbf{I}$ is an identity matrix, $\mathbf{L}_\mathcal{L}$ is the graph Laplacian computed based on $\mathbf{X}_\mathcal{L}$, which is a principal submatrix of $\mathbf{L}$.

In the following, we will present a deterministic out-of-sample error bound for LapRLS trained on the subsampled data, which is among the main contributions of this paper.

**Theorem 2.** *For any fixed $\mathbf{V} = [\mathbf{v}_1, \ldots, \mathbf{v}_m]$ and $\mathbf{X} = [\mathbf{x}_1, \ldots, \mathbf{x}_n]$, and a subsample $\mathcal{L}$ of $\mathbf{X}$, the expected error of LapRLS trained on $\mathcal{L}$ in predicting the true response $\mathbf{V}^T\mathbf{w}^*$ is upper bounded as*

$$\mathbb{E}||\mathbf{V}^T\hat{\mathbf{w}}_\mathcal{L} - \mathbf{V}^T\mathbf{w}^*||_2^2 \leq (B + \sigma^2)tr\left(\mathbf{V}^T(\mathbf{X}_\mathcal{L}\mathbf{X}_\mathcal{L}^T + \lambda_A \mathbf{I} + \lambda_I \mathbf{X}_\mathcal{L}\mathbf{L}_\mathcal{L}\mathbf{X}_\mathcal{L}^T)^{-1}\mathbf{V}\right). \tag{6}$$

*Proof.* Let $\mathbf{M}_\mathcal{L} = \lambda_A \mathbf{I} + \lambda_I \mathbf{X}_\mathcal{L}\mathbf{L}_\mathcal{L}\mathbf{X}_\mathcal{L}^T$. Given $\mathcal{L}$, the expected error (where the expectation is w.r.t. $\boldsymbol{\epsilon}_\mathcal{L}$) is given by

$$\mathbb{E}||\mathbf{V}^T\hat{\mathbf{w}}_\mathcal{L} - \mathbf{V}^T\mathbf{w}^*||_2^2$$
$$= \mathbb{E}||\mathbf{V}^T(\mathbf{X}_\mathcal{L}\mathbf{X}_\mathcal{L}^T + \mathbf{M}_\mathcal{L})^{-1}\mathbf{X}_\mathcal{L}\mathbf{y}_\mathcal{L} - \mathbf{V}^T\mathbf{w}^*||_2^2$$
$$= \underbrace{||\mathbf{V}^T(\mathbf{X}_\mathcal{L}\mathbf{X}_\mathcal{L}^T + \mathbf{M}_\mathcal{L})^{-1}\mathbf{X}_\mathcal{L}\mathbf{X}_\mathcal{L}^T\mathbf{w}^* - \mathbf{V}^T\mathbf{w}^*||_2^2}_{A_1} + \underbrace{\mathbb{E}||\mathbf{V}^T(\mathbf{X}_\mathcal{L}\mathbf{X}_\mathcal{L}^T + \mathbf{M}_\mathcal{L})^{-1}\mathbf{X}_\mathcal{L}\boldsymbol{\epsilon}_\mathcal{L}||_2^2}_{A_2}, \tag{7}$$

where the second equality follows from $\mathbf{y}_\mathcal{L} = \mathbf{X}_\mathcal{L}\mathbf{w}^* + \boldsymbol{\epsilon}_\mathcal{L}$. Now we bound the two terms in the right hand side respectively.

The first term is bounded by

$$A_1 = ||\mathbf{V}^T(\mathbf{X}_\mathcal{L}\mathbf{X}_\mathcal{L}^T + \mathbf{M}_\mathcal{L})^{-1}\mathbf{M}_\mathcal{L}\mathbf{w}^*||_2^2 \leq ||\mathbf{V}^T(\mathbf{X}_\mathcal{L}\mathbf{X}_\mathcal{L}^T + \mathbf{M}_\mathcal{L})^{-1}\mathbf{M}_\mathcal{L}^{\frac{1}{2}}||_F^2 ||\mathbf{M}_\mathcal{L}^{\frac{1}{2}}\mathbf{w}^*||_2^2$$
$$= B tr\left(\mathbf{V}^T(\mathbf{X}_\mathcal{L}\mathbf{X}_\mathcal{L}^T + \mathbf{M}_\mathcal{L})^{-1}\mathbf{M}_\mathcal{L}(\mathbf{X}_\mathcal{L}\mathbf{X}_\mathcal{L}^T + \mathbf{M}_\mathcal{L})^{-1}\mathbf{V}\right)$$
$$\leq B tr\left(\mathbf{V}^T(\mathbf{X}_\mathcal{L}\mathbf{X}_\mathcal{L}^T + \mathbf{M}_\mathcal{L})^{-1}\mathbf{V}\right) \tag{8}$$

where the first inequality is due to Cauchy Schwarz's inequality, and the second inequality follows from dropping the negative term.

Similarly, the second term can be bounded by

$$A_2 \leq \sigma^2 tr\left(\mathbf{V}^T(\mathbf{X}_\mathcal{L}\mathbf{X}_\mathcal{L}^T + \mathbf{M}_\mathcal{L})^{-1}\mathbf{X}_\mathcal{L}\mathbf{X}_\mathcal{L}^T(\mathbf{X}_\mathcal{L}\mathbf{X}_\mathcal{L}^T + \mathbf{M}_\mathcal{L})^{-1}\mathbf{V}\right)$$
$$\leq \sigma^2 tr\left(\mathbf{V}^T(\mathbf{X}_\mathcal{L}\mathbf{X}_\mathcal{L}^T + \mathbf{M}_\mathcal{L})^{-1}\mathbf{V}\right), \tag{9}$$

where the first equality uses $\mathbb{E}[\boldsymbol{\epsilon}_\mathcal{L}\boldsymbol{\epsilon}_\mathcal{L}^T] \leq \sigma^2 \mathbf{I}$, and it becomes equality if $\epsilon_i$ are independent and identically distributed (i.i.d.). Combing Eqs. (8) and (9) completes the proof. □

Note that in the above theorem, the sample $\mathbf{V}$ could be either the same as or different from the sample $\mathbf{X}$. Sometimes, we are also interested in the expected estimation error of $\mathbf{w}^*$ as follows.

**Theorem 3.** *For any fixed $\mathbf{X}$, and a subsample $\mathcal{L}$ of $\mathbf{X}$, the expected error of LapRLS trained on $\mathcal{L}$ in estimating the true weight vector $\mathbf{w}^*$ is upper bounded as*

$$\mathbb{E}||\hat{\mathbf{w}}_{\mathcal{L}} - \mathbf{w}^*||_2^2 \leq (B + \sigma^2)tr\left((\mathbf{X}_{\mathcal{L}}\mathbf{X}_{\mathcal{L}}^T + \lambda_A\mathbf{I} + \lambda_I\mathbf{X}_{\mathcal{L}}\mathbf{L}_{\mathcal{L}}\mathbf{X}_{\mathcal{L}}^T)^{-1}\right) \tag{10}$$

The proof of this theorem follows similar derivations of Theorem 2.

### 3.3 The Criterion of Selective Labeling

From Theorem 2, we can see that given a subsample $\mathcal{L}$ of $\mathbf{X}$, the expected prediction error of LapRLS on $\mathbf{V}$ is upper bounded by Eq. (6). In addition, the right hand side of Eq. (6) does not depend on the labels, i.e., $\mathbf{y}$. More importantly, the error bound derived in this paper is deterministic, which is unlike those probabilistic error bounds derived based on Rademacher complexity [3] or algorithmic stability [6]. Since those probabilistic error bounds only hold for i.i.d. sample rather than a particular sample, they cannot provide a criterion to choose a subsample set for labeling due to the correlation between the pool of candidate points and the i.i.d. sample. On the contrary, the deterministic error bound does not suffer from such a kind of problem. Therefore, it provides a natural criterion for selective labeling.

In detail, given a pool of candidate data points, i.e., $\mathbf{X}$, we propose to find a subsample $\mathcal{L}$ of $\{1, \ldots, n\}$, by minimizing the follow objective function

$$\arg\min_{\mathcal{L}\subset\{1,\ldots,n\}} \text{tr}\left(\mathbf{X}^T(\mathbf{X}_{\mathcal{L}}\mathbf{X}_{\mathcal{L}}^T + \lambda_I\mathbf{X}_{\mathcal{L}}\mathbf{L}_{\mathcal{L}}\mathbf{X}_{\mathcal{L}}^T + \lambda_A\mathbf{I})^{-1}\mathbf{X}\right), \tag{11}$$

where we simply assume $\mathbf{V} = \mathbf{X}$. The above problem is a combinatorial optimization problem. Finding the global optimal solution is NP-hard. One potential way to solve it is greedy forward (or backward) selection. However, it is inefficient. Here we propose an efficient algorithm, which solves its continuous relaxation.

### 3.4 Reformulation

We introduce a selection matrix $\mathbf{S} \in \mathbb{R}^{n \times l}$, which is defined as

$$S_{ij} = \begin{cases} 1, & \text{if } \mathbf{x}_i \text{ is selected as the } j\text{-point in } \mathcal{L} \\ 0, & \text{otherwise.} \end{cases} \tag{12}$$

It is easy to check that each column of $\mathbf{S}$ has one and only one 1, and each row has at most one 1. The constraint set for $\mathbf{S}$ can be defined as

$$\mathcal{S}_1 = \{\mathbf{S}|\mathbf{S} \in \{0,1\}^{n \times l}, \mathbf{S}^T\mathbf{1} = \mathbf{1}, \mathbf{S}\mathbf{1} \leq \mathbf{1}\}, \tag{13}$$

where $\mathbf{1}$ is a vector of all ones, or equivalently,

$$\mathcal{S}_2 = \{\mathbf{S}|\mathbf{S} \in \{0,1\}^{n \times l}, \mathbf{S}^T\mathbf{S} = \mathbf{I}\}, \tag{14}$$

where $\mathbf{I}$ is an identity matrix.

Based on $\mathbf{S}$, we have $\mathbf{X}_{\mathcal{L}} = \mathbf{X}\mathbf{S}$ and $\mathbf{L}_{\mathcal{L}} = \mathbf{S}^T\mathbf{L}\mathbf{S}$. Thus, Eq. (11) can be equivalently reformulated as

$$\arg\min_{\mathbf{S}\in\mathcal{S}_2} \text{tr}\left(\mathbf{X}^T(\mathbf{X}\mathbf{S}\mathbf{S}^T\mathbf{X}^T + \lambda_I\mathbf{X}\mathbf{S}\mathbf{S}^T\mathbf{L}\mathbf{S}\mathbf{S}^T\mathbf{X}^T + \lambda_A\mathbf{I})^{-1}\mathbf{X}\right)$$

$$= \arg\min_{\mathbf{S}\in\mathcal{S}_2} \text{tr}\left(\mathbf{X}^T(\mathbf{X}\mathbf{S}\mathbf{S}^T\mathbf{L}'\mathbf{S}\mathbf{S}^T\mathbf{X}^T + \lambda_A\mathbf{I})^{-1}\mathbf{X}\right), \tag{15}$$

where $\mathbf{L}' = \mathbf{I} + \lambda_I\mathbf{L}$. The above optimization problem is still a discrete optimization. Let

$$\mathcal{S}_3 = \{\mathbf{S}|\mathbf{S} \geq 0, \mathbf{S}^T\mathbf{S} = \mathbf{I}\}, \tag{16}$$

where we relax the binary constraint on $\mathbf{S}$ into nonnegative constraint. Note that $\mathcal{S}_3$ is a matching polytope [17]. Then we solve the following continuous optimization,

$$\arg\min_{\mathbf{S}\in\mathcal{S}_3} \text{tr}\left(\mathbf{X}^T(\mathbf{X}\mathbf{S}\mathbf{S}^T\mathbf{L}'\mathbf{S}\mathbf{S}^T\mathbf{X}^T + \lambda_A\mathbf{I})^{-1}\mathbf{X}\right). \tag{17}$$

We derive a projected gradient descent algorithm to find a local optimum of Eq. (17). We first ignore the nonnegative constraint on $\mathbf{S}$. Since $\mathbf{S}^T\mathbf{S} = \mathbf{I}$, we introduce a Lagrange multiplier $\mathbf{\Lambda} \in \mathbb{R}^{l \times l}$, thus the Lagrangian function is

$$L(\mathbf{S}) = \text{tr}\left(\mathbf{X}^T(\mathbf{X}\mathbf{S}\mathbf{S}^T\mathbf{L}'\mathbf{S}\mathbf{S}^T\mathbf{X}^T + \lambda_A\mathbf{I})^{-1}\mathbf{X}\right) + \text{tr}\left(\mathbf{\Lambda}(\mathbf{S}^T\mathbf{S} - \mathbf{I})\right). \tag{18}$$

The derivative of $L(\mathbf{S})$ with respect to $\mathbf{S}$ is[2]

$$\frac{\partial L}{\partial \mathbf{S}} = -2(\mathbf{X}^T\mathbf{B}\mathbf{X}\mathbf{S}\mathbf{S}^T\mathbf{L}'\mathbf{S} + \mathbf{L}'\mathbf{S}\mathbf{S}^T\mathbf{X}^T\mathbf{B}\mathbf{X}\mathbf{S}) + 2\mathbf{S}\mathbf{\Lambda}, \tag{19}$$

where $\mathbf{B} = \mathbf{A}^{-1}(\mathbf{X}\mathbf{X}^T)\mathbf{A}^{-1}$ and $\mathbf{A} = \mathbf{X}\mathbf{S}\mathbf{S}^T\mathbf{L}'\mathbf{S}\mathbf{S}^T\mathbf{X}^T + \lambda\mathbf{I}$. Note that the computational burden of the derivative is $\mathbf{A}^{-1}$, which is the inverse of a $d \times d$ matrix. To overcome this problem, we use the Woodbury matrix identity [12]. Then $\mathbf{A}^{-1}$ can be computed as

$$\mathbf{A}^{-1} = \frac{1}{\lambda}\mathbf{I} - \frac{1}{\lambda^2}\mathbf{X}\mathbf{S}\left((\mathbf{S}^T\mathbf{L}'\mathbf{S})^{-1} + \frac{1}{\lambda}\mathbf{S}^T\mathbf{X}^T\mathbf{X}\mathbf{S}\right)^{-1}\mathbf{S}^T\mathbf{X}^T, \tag{20}$$

where $\mathbf{S}^T\mathbf{L}'\mathbf{S}$ is a $l \times l$ matrix, whose inverse can be solved efficiently when $l \ll d$.

To determine the Lagrange multiplier $\mathbf{\Lambda}$, left multiplying Eq. (19) by $\mathbf{S}^T$, and using the fact that $\mathbf{S}^T\mathbf{S} = \mathbf{I}$, we obtain

$$\mathbf{\Lambda} = \mathbf{S}^T\mathbf{X}^T\mathbf{B}\mathbf{X}\mathbf{S}\mathbf{S}^T\mathbf{L}'\mathbf{S} + \mathbf{S}^T\mathbf{L}'\mathbf{S}\mathbf{S}^T\mathbf{X}^T\mathbf{B}\mathbf{X}\mathbf{S}. \tag{21}$$

Substituting the Lagrange multiplier $\mathbf{\Lambda}$ back into Eq. (19), we can obtain the derivative depending only on $\mathbf{S}$. Thus we can use projected gradient descent to find a local optimal solution for Eq. (17). In each iteration, it takes a step proportional to the negative of the gradient of the function at the current point, followed by a projection back into the nonnegative set.

### 3.5 Discretization

Till now, we have obtained a local optimal solution $\mathbf{S}^*$ by projected gradient descent. However, this $\mathbf{S}^*$ contains continuous values. In other words, $\mathbf{S}^* \in \mathcal{S}_3$. In order to determine which $l$ data points to select, we need to project $\mathbf{S}^*$ into $\mathcal{S}_1$. We use a simple greedy procedure to conduct the discretization: we first find the largest element in $\mathbf{S}$ (if there exist multiple largest elements, we choose any one of them), and mark its row and column; then from the unmarked columns and rows we find the largest element and also mark it; this procedure is repeated until we find $l$ elements.

## 4   Related Work

We notice that our proposed method shares similar spirit with *optimal experimental design*[3] in statistics [1, 20, 16], whose intent is to select the most informative data points to learn a function which has minimum variance of estimation, or minimum variance of prediction.

For example, *A-Optimal Design* (AOD) minimizes the expected variance of the model parameter. In particular, for ridge regression, it optimizes the following criterion,

$$\arg\min_{\mathcal{L} \subset \{1,...,n\}} \text{tr}\left((\mathbf{X}_\mathcal{L}\mathbf{X}_\mathcal{L}^T + \lambda_A\mathbf{I})^{-1}\right), \tag{22}$$

where $\mathbf{I}$ is an identity matrix. We can recover this criterion by setting $\lambda_I = 0$ in Theorem 3. However, the pitfall of AOD is that it does not characterize the quality of predictions on the data, which is essential for classification or regression.

To overcome the shortcoming of A-optimal design, Yu et al. [20] proposed a *Transdutive Experimental Design* (TED) approach. TED selects the samples which minimize the expected predictive variance of ridge regression on the data,

$$\arg\min_{\mathcal{L} \subset \{1,...,n\}} \text{tr}\left(\mathbf{X}^T(\mathbf{X}_\mathcal{L}\mathbf{X}_\mathcal{L}^T + \lambda_A\mathbf{I})^{-1}\mathbf{X}\right). \tag{23}$$

Although TED is motivated by minimizing the variance of the prediction, it is very interesting to demonstrate that the above criterion is coinciding with minimizing the out-of-sample error bound in Theorem 2 with $\lambda_I = 0$. The reason is that for ridge regression, the upper bounds of the bias and variance terms share a common factor tr $\left( \mathbf{X}^T (\mathbf{X}_\mathcal{L} \mathbf{X}_\mathcal{L}^T + \lambda_A \mathbf{I})^{-1} \mathbf{X} \right)$. This is a very important observation because it explains why TED performs very well even though its criterion is minimizing the variance of the prediction. Furthermore, TED can be seen as a special case of our proposed method.

He et al. [16] proposed *Laplacian Optimal Design* (LOD), which selects data points that minimize the expected predictive variance of Laplacian regularized least squares [4] on the data,

$$\arg \min_{\mathcal{L} \subset \{1,...,n\}} \text{tr} \left( \mathbf{X}^T (\lambda_I \mathbf{X}\mathbf{L}\mathbf{X}^T + \mathbf{X}_\mathcal{L} \mathbf{X}_\mathcal{L}^T + \lambda_A \mathbf{I})^{-1} \mathbf{X} \right), \tag{24}$$

where the graph Laplacian $\mathbf{L}$ is computed on all the data points in the pool, i.e., $\mathbf{X}$. LOD selects the points by $\mathbf{X}_\mathcal{L} \mathbf{X}_\mathcal{L}^T$ while leaving the graph Laplacian term $\mathbf{X}\mathbf{L}\mathbf{X}^T$ fixed. However, our method selects the points by $\mathbf{X}_\mathcal{L} \mathbf{X}_\mathcal{L}^T$ as well as the graph Laplacian term i.e., $\mathbf{X}_\mathcal{L} \mathbf{L}_\mathcal{L} \mathbf{X}_\mathcal{L}^T$. This difference is essential, because our criterion has a strong theoretical foundation, i.e., minimizing the out-of-sample error bound of LapRLS. This explains the non-significant improvement of LOD over TED. Admittedly, the term $\mathbf{X}_\mathcal{L} \mathbf{L}_\mathcal{L} \mathbf{X}_\mathcal{L}^T$ in our method raised a challenge for optimization. Yet it has been well-solved by the projected gradient descent algorithm derived in previous section.

We also notice that similar problem was studied for graphs [13]. However, their method cannot be applied to our setting, because their input is restricted to the adjacency matrix of a graph.

## 5 Experiments

In this section, we evaluate the proposed method on both synthetic and real-world datasets, and compare it with the state-of-the-art methods. All the experiments are conducted in Matlab.

### 5.1 Compared Methods

To demonstrate the effectiveness of our proposed method, we compare it with the following baseline approaches: Random Sampling (**Random**) uniformly selects data points from the pool as training data. It is the simplest baseline for label selection. A-Optimal Design (**AOD**) is a classic experimental design method proposed in the community of statistics. There is a parameter $\lambda_A$ to be tuned. Transductive Experiment Design (**TED**) is proposed in [20], which is the state-of-the-art (non-adaptive) active learning method. There is a parameter $\lambda_A$ to be tuned. Laplacian Optimal Design (**LOD**) [16] is an extension of TED, which incorporates the manifold structure of the data. Selective Labeling via Error Bound Minimization (**Bound**) is the proposed method. There are two tunable parameters $\lambda_A$ and $\lambda_I$ in both LOD and Bound.

Both LOD and Bound use graph Laplacian. To compute it, we first normalize each data point into a vector with unit $\ell_2$-norm. Then we construct a $5$-NN graph and use the cosine distance to measure the similarity between data points throughout of our experiments.

Note that the problem setting of our study is to select a batch of data points to label without training a classifier. Therefore, we do not compare our method with typical active learning methods such as SVM active learning [19, 18] and agnostic active learning [2].

After selecting the data points by the above methods, we train a LapRLS [4] as the learner to do classification. There are two parameters in LapRLS, i.e., $\lambda_A$ and $\lambda_I$.

### 5.2 Synthetic Dataset

To get an intuitive picture of how the above methods (except random sampling, which is trivial) work differently, we show their experimental results on a synthetic dataset in Figure 1. This dataset contains two circles, each of which constitutes a class. It has strong manifold structure. We let the compared methods select $8$ data points. As can be seen, the data points selected by AOD are concentrated on the inner circle (belonging to one class), which are not able to train a classifier. The data points selected by TED, LapIOD and Bound are distributed on both inner and outer circles

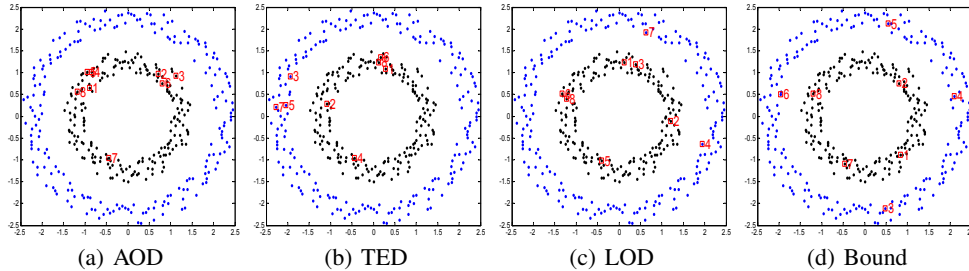

|        |        |        |        |
|--------|--------|--------|--------|
| (a) AOD | (b) TED | (c) LOD | (d) Bound |

Figure 1: Selected points (the red marks) on the two circles dataset by (a) AOD; (b) TED; (c) LOD; and (d) Bound.

(belonging to different classes), which are good at training a learner. Furthermore, the 8 data points selected by Bound are uniformly distributed on the two circles, four from the inner circle, and the other four from the outer circle, which can better represent the original data.

## 5.3 Real Datasets & Parameter Settings

In the following, we use three real-world benchmark datasets to evaluate the compared methods.

**wdbc** is the Wisconsin Diagnostic Breast Cancer data set, which is from UCI machine learning repository[4]. It aims at predicting the breast cancer as benign or malignant based on the digitalized images. There are 357 positive samples and 212 negative samples. Each sample has 32 attributes.

**ORL** face database[5] contains 10 images for each of the 40 human subjects, which were taken at different times, varying the lighting, facial expressions and facial details. The original images (with 256 gray levels) have size $92 \times 112$, which are resized to $32 \times 32$ for efficiency.

**Isolet** was first used in [11]. It contains 150 people who spoke each letter of the alphabet twice. The speakers are grouped into sets of 30 speakers each, and we use the first group, referred to Isolet1. Each sample is represented by a 617-dimensional feature vector.

For each data set, we randomly select $20\%$ data as held-out set for model selection, and the rest $80\%$ data as work set. In order to randomize the experiments, in each run of experiments, we restrict the training data (pool of candidate data points) to be selected from a random sampling of $50\%$ work set (which accounts for $40\%$ of the total data). The remaining half data ($40\%$ of the total data) is used as test set. Once the labeled data are selected, we train a semi-supervised version of LapRLS, which uses both labeled and unlabeled data (all the training data) for manifold regularization. We report the classification result on the test set. This random split was repeated 10 times, thus we can compute the mean and standard deviation of the classification accuracy.

The parameters of compared methods (See Section 5.1) are tuned by 2-fold cross validation on the held-out set. For the parameters of LapRLS, we use the same parameters of LOD (or Bound) for LapRLS. For the wdbc dataset, the chosen parameters are $\lambda_A = 0.001, \lambda_I = 0.01$. For ORL, $\lambda_A = 0.0001, \lambda_I = 0.001$. For Isolet1, $\lambda_A = 0.01, \lambda_I = 0.001$.

For wdbc, we let the compared methods incrementally choose $\{2, 4, \ldots, 20\}$ points to label, for ORL, we incrementally choose $\{80, 90, \ldots, 150\}$ points for labeling, and for Isolet1, we choose $\{30, 40, \ldots, 120\}$ points to query.

## 5.4 Results on Real Datasets

The experimental results are shown in Figure 2. In all subfigures, the x-axis represents the number of labeled points, while the y-axis is the averaged classification accuracy on the test data over 10 runs. In order to show some concrete results, we also list the accuracy and running time (in second) of all the compared methods on the three datasets with 2, 80 and 30 labeled data points respectively in

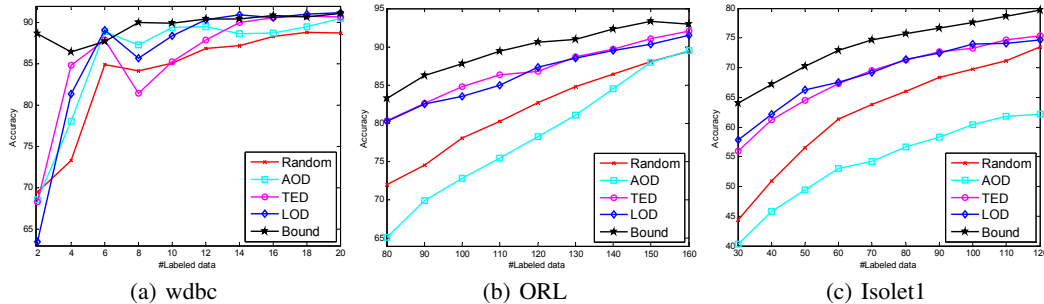

|  | | |
|---|---|---|
| (a) wdbc | (b) ORL | (c) Isolet1 |

Figure 2: Comparison of different methods on (a) wdbc; (b) ORL; and (c) Isolet1 using LapRLS.

Table 1: Classification accuracy (%) and running time (in second) of compared methods on the three datasets.

| Dataset | wdbc (2 labeled) | | ORL (80 labeled) | | Isolet1 (30 labeled) | |
|---|---|---|---|---|---|---|
|  | Acc | time | Acc | time | Acc | time |
| Random | 69.47±14.56 | – | 72.00±4.05 | – | 44.36±3.09 | – |
| AOD | 68.59±12.46 | 0.0 | 65.17±3.14 | 32.2 | 40.27±2.24 | 7.4 |
| TED | 68.33±10.68 | 0.0 | 80.33±2.94 | 39.6 | 55.98±2.54 | 41.1 |
| LOD | 63.48±8.38 | 0.1 | 80.25±2.64 | 41.7 | 57.79±1.87 | 41.5 |
| Bound | **88.68±2.82** | 0.3 | **83.25±3.17** | 23.4 | **61.99±2.14** | 17.4 |

Table 1. For each dataset, we did paired t-tests between the proposed method and the other methods in the 95% confidence interval. If it is significant over all the other methods, the corresponding entry of Bound is bolded.

We observe that the proposed selective labeling method greatly outperforms the other methods at most cases. AOD is usually worse than random sampling. The reason is that minimizing the variance of model parameter does not guarantee the quality of predictions on the data. TED performs very well. As we mentioned before, the criterion of TED coincides with minimizing the out-of-sample error bound of ridge regression. This explains its good empirical performance. The performance of LOD is slightly better than TED. This is because LOD incorporates the geometric structure into TED. The superior performance of our method is attributed to its theoretical foundation, which guarantees that the learner (LapRLS) can achieve small error on the test data. In addition, the running time of our method is comparable to or even less than the running time of the other methods.

One may argue that the above comparison is not fair because we use LapRLS as the learner, which tends to fit the proposed method. Therefore, we also compare different methods using ridge regression (RR) as the learner. We find that our proposed method is also much better than the other methods using RR. For the space limit, we omit the results here and put them in the supplemental material.

# 6 Conclusions

The main contributions of this paper are: (1) We present a deterministic out-of-sample error bound for LapRLS; (2) we present a selective labeling method by minimizing this upper bound; and (3) we present a simple yet effective algorithm to optimize the criterion for selective labeling.

# Acknowledgement

The work was supported in part by U.S. National Science Foundation grants IIS-0905215, CNS-0931975, the U.S. Army Research Laboratory under Cooperative Agreement No. W911NF-09-2-0053 (NS-CTA), the U.S. Air Force Office of Scientific Research MURI award FA9550-08-1-0265, and MIAS, a DHS-IDS Center for Multimodal Information Access and Synthesis at UIUC. We would like to thank the anonymous reviewers for their helpful comments.

## Footnotes

[1]https://www.mturk.com/

[2]The calculation of the derivative is non-trivial, please refer to the supplementary material for detail.

[3]Some literature also call it active learning, while our understand is there is no adaptive interaction between the learner and the oracle within optimal experimental design. Therefore, it is better to call it *nonadaptive active learning*.

[4]http://archive.ics.uci.edu/ml/

[5]http://www.cl.cam.ac.uk/Research/DTG/attarchive:pub/data

# References

[1] A. D. Anthony Atkinson and R. Tobias. *Optimum Experimental Designs*. Oxford University Press, 2007.

[2] M.-F. Balcan, A. Beygelzimer, and J. Langford. Agnostic active learning. In *ICML*, pages 65–72, 2006.

[3] P. L. Bartlett and S. Mendelson. Rademacher and gaussian complexities: Risk bounds and structural results. *Journal of Machine Learning Research*, 3:463–482, 2002.

[4] M. Belkin, P. Niyogi, and V. Sindhwani. Manifold regularization: A geometric framework for learning from labeled and unlabeled examples. *Journal of Machine Learning Research*, 7:2399–2434, 2006.

[5] A. Beygelzimer, D. Hsu, J. Langford, and T. Zhang. Agnostic active learning without constraints. In *NIPS*, pages 199–207, 2010.

[6] O. Bousquet and A. Elisseeff. Stability and generalization. *Journal of Machine Learning Research*, 2:499–526, 2002.

[7] O. Chapelle, B. Schölkopf, and A. Zien, editors. *Semi-Supervised Learning*. MIT Press, Cambridge, MA, 2006.

[8] F. R. K. Chung. *Spectral Graph Theory*. American Mathematical Society, February 1997.

[9] D. A. Cohn, L. E. Atlas, and R. E. Ladner. Improving generalization with active learning. *Machine Learning*, 15(2):201–221, 1994.

[10] D. A. Cohn, Z. Ghahramani, and M. I. Jordan. Active learning with statistical models. In *NIPS*, pages 705–712, 1994.

[11] M. A. Fanty and R. A. Cole. Spoken letter recognition. In *NIPS*, pages 220–226, 1990.

[12] G. H. Golub and C. F. V. Loan. *Matrix computations (3rd ed.)*. Johns Hopkins University Press, Baltimore, MD, USA, 1996.

[13] A. Guillory and J. Bilmes. Active semi-supervised learning using submodular functions. In *UAI*, pages 274–282, 2011.

[14] S. Hanneke. Rates of convergence in active learning. *The Annals of Statistics*, 39(1):333–361, 2011.

[15] T. Hastie, R. Tibshirani, and J. H. Friedman. *The elements of statistical learning: data mining, inference, and prediction*. New York: Springer-Verlag, 2001.

[16] X. He, W. Min, D. Cai, and K. Zhou. Laplacian optimal design for image retrieval. In *SIGIR*, pages 119–126, 2007.

[17] B. Korte and J. Vygen. *Combinatorial Optimization: Theory and Algorithms*. Springer Publishing Company, Incorporated, 4th edition, 2007.

[18] G. Schohn and D. Cohn. Less is more: Active learning with support vector machines. In *ICML*, pages 839–846, 2000.

[19] S. Tong and D. Koller. Support vector machine active learning with applications to text classification. In *ICML*, pages 999–1006, 2000.

[20] K. Yu, J. Bi, and V. Tresp. Active learning via transductive experimental design. In *ICML*, pages 1081–1088, 2006.

[21] D. Zhou, O. Bousquet, T. N. Lal, J. Weston, and B. Schölkopf. Learning with local and global consistency. In *NIPS*, 2003.

[22] X. Zhu, Z. Ghahramani, and J. D. Lafferty. Semi-supervised learning using gaussian fields and harmonic functions. In *ICML*, pages 912–919, 2003.

